# Globally Convergent Dual MAP LP Relaxation Solvers using Fenchel-Young Margins

**Alexander G. Schwing**
ETH Zurich
aschwing@inf.ethz.ch

**Tamir Hazan**
TTI Chicago
tamir@ttic.edu

**Marc Pollefeys**
ETH Zurich
pomarc@inf.ethz.ch

**Raquel Urtasun**
TTI Chicago
rurtasun@ttic.edu

## Abstract

While finding the exact solution for the MAP inference problem is intractable for many real-world tasks, MAP LP relaxations have been shown to be very effective in practice. However, the most efficient methods that perform block coordinate descent can get stuck in sub-optimal points as they are not globally convergent. In this work we propose to augment these algorithms with an $\epsilon$-descent approach and present a method to efficiently optimize for a descent direction in the sub-differential using a margin-based formulation of the Fenchel-Young duality theorem. Furthermore, the presented approach provides a methodology to construct a primal optimal solution from its dual optimal counterpart. We demonstrate the efficiency of the presented approach on spin glass models and protein interaction problems and show that our approach outperforms state-of-the-art solvers.

## 1   Introduction

Graphical models are a common method to describe the dependencies of a joint probability distribution over a set of discrete random variables. Finding the most likely configuration of a distribution defined by such a model, *i.e.*, the maximum a-posteriori (MAP) assignment, is one of the most important inference tasks. Unfortunately, it is a computationally hard problem for many interesting applications. However, it has been shown that linear programming (LP) relaxations recover the MAP assignment in many cases of interest (*e.g.*, [13, 23]).

Due to the large amount of variables and constraints, solving inference problems in practice still remains a challenge for standard LP solvers. Development of specifically tailored algorithms has since become a growing area of research. Many of these designed solvers consider the dual program, thus they are based on local updates that follow the graphical model structure, which ensures suitability for very large problems. Unfortunately, the dual program is non-smooth, hence introducing difficulties to existing solvers. For example, block coordinate descent algorithms, typically referred to as convex max-product, monotonically decrease the dual objective and converge very fast, but are not guaranteed to reach the global optimum of the dual program [3, 6, 11, 14, 17, 20, 22, 24, 25]. Different approaches to overcome the sub-optimality of the convex max-product introduced different perturbed programs for which convergence to the dual optimum is guaranteed, *e.g.*, smoothing, proximal methods and augmented Lagrangian methods [6, 7, 8, 16, 18, 19, 27]. However, since these algorithms consider a perturbed program they are typically slower than the convex max-product variants [8, 18].

In this work we propose to augment the convex max-product algorithm with a steepest $\epsilon$-descent approach to monotonically decrease the dual objective until reaching the global optimum of the dual program. To perform the $\epsilon$-descent we explore the $\epsilon$-subgradients of the dual program, and provide a method to search for a descent direction in the $\epsilon$-subdifferential using a margin-based formulation of the Fenchel-Young duality theorem. This characterization also provides a new algorithm to

construct a primal optimal solution for the LP relaxation from a dual optimal solution. We demonstrate the effectiveness of our approach on spin glass models and protein-protein interactions taken from the probabilistic inference challenge (PIC 2011)[1]. We illustrate that the method exhibits nice convergence properties while possessing optimality certificates.

We begin by introducing the notation, MAP LP relaxations and their dual programs. We subsequently describe the subgradients of the dual and provide an efficient procedure to recover a primal optimal solution. We explore the $\epsilon$-subgradients of the dual objective, and introduce an efficient globally convergent dual solver based on the $\epsilon$-margin of the Fenchel-Young duality theorem. Finally, we extend our approach to graphical models over general region graphs.

## 2  Background

Graphical models encode joint distributions over discrete product spaces $X = X_1 \times \cdots \times X_n$. The joint probability is defined by combining energy functions over subsets of variables. Throughout this work we consider two types of functions: single variable functions, $\theta_i(x_i)$, which correspond to the $n$ vertices in the graph, $i \in \{1, ..., n\}$, and functions over subsets of variables $\theta_\alpha(x_\alpha)$, for $\alpha \subset \{1, .., n\}$, that correspond to the graph hyperedges. The joint distribution is then given by $p(x) \propto \exp(\sum_{i \in V} \theta_i(x_i) + \sum_{\alpha \in E} \theta_\alpha(x_\alpha))$. In this paper we focus on estimating the MAP, *i.e.*, finding the assignment that maximizes the probability, or equivalently minimizes the energy which is the negative log probability. Estimating the MAP can be written as a program of the form [10]:

$$\underset{x_1, ..., x_n}{\operatorname{argmax}} \sum_{i \in V} \theta_i(x_i) + \sum_{\alpha \in E} \theta_\alpha(x_\alpha). \tag{1}$$

Due to its combinatorial nature, this problem is NP-hard for general graphical models. It is tractable only in some special cases such as tree structured graphs, where specialized dynamic programming algorithms (*e.g.*, max-product belief propagation) are guaranteed to recover the optimum.

The MAP program in Eq. (1) has a linear form, thus it is naturally represented as an integer linear program. Its tractable relaxation is obtained by replacing the integral constraints with non-negativity constraints as follows:

$$\max_{b_i, b_\alpha} \sum_{\alpha, x_\alpha} b_\alpha(x_\alpha)\theta_\alpha(x_\alpha) + \sum_{i, x_i} b_i(x_i)\theta_i(x_i) \tag{2}$$

$$\text{s.t.} \quad b_i(x_i), b_\alpha(x_\alpha) \geq 0, \ \sum_{x_\alpha} b_\alpha(x_\alpha) = 1, \sum_{x_i} b_i(x_i) = 1, \ \sum_{x_\alpha \setminus x_i} b_\alpha(x_\alpha) = b_i(x_i).$$

Whenever the maximizing argument to above linear program happens to be integral, *i.e.*, the optimal beliefs satisfy $b_i(x_i), b_\alpha(x_\alpha) \in \{0, 1\}$, the program value equals the MAP value. Moreover, the maximum arguments of the optimal beliefs point toward the MAP assignment [26].

We denote by $N(i)$ the edges that contain vertex $i$ and by $N(\alpha)$ the vertices in the edge $\alpha$. Following [22, 27] we consider the re-parametrized dual

$$q(\lambda) = \sum_i \max_{x_i} \left\{ \theta_i(x_i) + \sum_{\alpha \in N(i)} \lambda_{i \to \alpha}(x_i) \right\} + \sum_\alpha \max_{x_\alpha} \left\{ \theta_\alpha(x_\alpha) - \sum_{i \in N(\alpha)} \lambda_{i \to \alpha}(x_i) \right\}. \tag{3}$$

The dual program value upper bounds the primal program described in Eq. (2). Therefore to compute the primal optimal value one can minimize the dual upper bound. Using block coordinate descent on the dual objective amounts to optimizing blocks of dual variables while holding the remaining ones fixed. This results in the convex max-product message-passing update rules [6, 17]:

Repeat until convergence, for every $i = 1, ..., n$:

$$\forall x_i, \alpha \in N(i) \quad \mu_{\alpha \to i}(x_i) = \max_{x_\alpha \setminus x_i} \left\{ \theta_\alpha(x_\alpha) + \sum_{j \in N(\alpha) \setminus i} \lambda_{j \to \alpha}(x_j) \right\}$$

$$\forall x_i, \alpha \in N(i) \quad \lambda_{i \to \alpha}(x_i) = \frac{1}{1 + |N(i)|} \left( \theta_i(x_i) + \sum_{\beta \in N(i)} \mu_{\beta \to i}(x_i) \right) - \mu_{\alpha \to i}(x_i)$$

[1]http://www.cs.huji.ac.il/project/PASCAL/index.php

The convex max-product algorithm is guaranteed to converge since it minimizes the dual function, which is lower bounded by the primal program. Interestingly, the convex max-product shares the same complexity as the max-product belief propagation, which is attained by replacing the coefficient $1/(1 + |N(i)|)$ by 1. It has, however, two fundamental problems. First, it can get stuck in non-optimal stationary points. This happens since the dual objective is non-smooth, thus the algorithm can reach a corner, for which the dual objective stays fixed when changing only a few variables. For example, consider the case of a minimization problem where we try to descend from a pyramid while taking only horizontal and vertical paths. We eventually stay at the same height. The second drawback of convex max-product is that it does not always produce a primal optimal solution, $b_i(x_i), b_\alpha(x_\alpha)$, even when it reaches a dual optimal solution.

In the next section, we consider the dual subgradients, and provide an efficient algorithm for detecting corners, as well as for decoding a primal optimal solution from a dual optimal solution. This is an intermediate step which facilitates the margin analysis of the Fenchel-Young duality theorem in Sec. 4. It provides an efficient way to get out of corners, and to reach the optimal dual value.

## 3   The Subgradients of the Dual Objective and Steepest Descent

Subgradients are generalizations of gradients for non-smooth convex functions. Consider the function $q(\lambda)$ in Eq. (3). A vector $d$ is called a subgradient of $q(\lambda)$ if it supports the epigraph of $q(\lambda)$ at $\lambda$, *i.e.*,

$$\forall \hat{\lambda} \quad q(\hat{\lambda}) - d^\top \hat{\lambda} \geq q(\lambda) - d^\top \lambda. \tag{4}$$

The supporting hyperplane at $(\lambda, q(\lambda))$ with slope $d$ takes the form $d^\top \lambda - q^*(d)$, when defining the conjugate dual as $q^*(d) = \max_\lambda \{d^\top \lambda - q(\lambda)\}$. From the definition of $q^*(d)$ one can derive the Fenchel-Young duality theorem: $q(\lambda) + q^*(d) \geq d^\top \lambda$, where equality holds if and only if $d$ is a supporting hyperplane at $(\lambda, q(\lambda))$. The set of all subgradients is called the subdifferential, denoted by $\partial q(\lambda)$, which can be characterized using the Fenchel-Young theorem as $\partial q(\lambda) = \{d : q(\lambda) + q^*(d) = \lambda^\top d\}$. The subdifferential provides a way to reason about the optimal solutions of $q(\lambda)$. Using Eq. (4) we can verify that $\lambda$ is dual optimal if and only if $0 \in \partial q(\lambda)$. In the following claim we characterize the subdifferential of the dual function $q(\lambda)$ using the Fenchel-Young duality theorem:

**Claim 1.** *Consider the dual function $q(\lambda)$ given in Eq. (3). Let $X_i^* = \operatorname{argmax}_{x_i}\{\theta_i(x_i) - \sum_{\alpha \in N(i)} \lambda_{i \to \alpha}(x_i)\}$ and $X_\alpha^* = \operatorname{argmax}_{x_\alpha}\{\theta_\alpha(x_\alpha) + \sum_{i \in N(\alpha)} \lambda_{i \to \alpha}(x_i)\}$. Then $d \in \partial q(\lambda)$, if and only if $d_{i \to \alpha}(x_i) = \sum_{x_\alpha \setminus x_i} b_\alpha(x_\alpha) - b_i(x_i)$ for probability distributions $b_i(x_i), b_\alpha(x_\alpha)$ whose nonzero entries belong to $X_i^*, X_\alpha^*$ respectively.*

**Proof:** Using the Fenchel-Young characterization of Eq. (4) for the max-function we obtain the set of maximizing elements $X_i^*, X_\alpha^*$. Summing over all regions $r \in \{i, \alpha\}$ while noticing the change of sign, we obtain the marginalization disagreements $d_{i \to \alpha}(x_i)$. □

The convex max-product algorithm performs block coordinate descent updates. Thus it iterates over vertices $i$ and computes optimal solutions $\lambda_{i \to \alpha}(x_i)$ for every $x_i, \alpha \in N(i)$ analytically, while holding the rest of the variables fixed. The claim above implies that the convex max-product iterates over $i$ and generates beliefs $b_i(x_i), b_\alpha(x_\alpha)$ for every $x_i, \alpha \in N(i)$ that agree on their marginal probabilities. This interpretation provides an insight into the non-optimal stationary points of the convex max-product, *i.e.*, points for which it is not able to generate consistent beliefs $b_\alpha(x_\alpha)$ when it iterates over $i = 1, \ldots, n$. The representation of the subdifferential as the amount of disagreement between the marginalization constraints provides a simple procedure to verify dual optimality, as well as to construct primal optimal solutions. This is summarized in the corollary below.

**Corollary 1.** *Given a point $\lambda$, and sets $X_i^*, X_\alpha^*$ as defined in Claim 1, let $x_i^*, x_\alpha^*$ be elements in $X_i^*, X_\alpha^*$ respectively. Consider the quadratic program*

$$\min_{b_i, b_\alpha} \sum_{i, x_i^*, \alpha \in N(i)} \left( \sum_{x_\alpha^* \setminus x_i^*} b_\alpha(x_\alpha^*) - b_i(x_i^*) \right)^2$$

$$s.t. \quad b_i(x_i^*), b_\alpha(x_\alpha^*) \geq 0, \quad \sum_{x_\alpha^*} b_\alpha(x_\alpha^*) = 1, \sum_{x_i^*} b_i(x_i^*) = 1.$$

*$\lambda$ is a dual optimal solution if and only if the value of the above program equals zero. Moreover, if $\lambda$ is a dual optimal solution, then the optimal beliefs $b_\alpha^*(x_\alpha), b_i^*(x_i)$ are also the optimal solution*

*of the primal program in Eq.* (2). *However, if $\lambda$ is not dual optimal, then the vector* $d_{i \to \alpha}^*(x_i) = \sum_{x_\alpha^* \backslash x_i^*} b_\alpha^*(x_\alpha^*) - b_i^*(x_i^*)$ *points towards the steepest descent direction of the dual function,* i.e.,

$$d^* = \underset{\|d\| \leq 1}{argmin} \quad \lim_{\alpha \downarrow 0} \frac{q(\lambda + \alpha d) - q(\lambda)}{\alpha}.$$

**Proof:** The steepest descent direction $d$ of $q$ is given by minimizing the directional derivative $q_d'$,

$$\min_{\|d\| \leq 1} q_d'(\lambda) = \min_{\|d\| \leq 1} \max_{y \in \partial q} d^\top y = \max_{y \in \partial q} \min_{\|d\| \leq 1} d^\top y = \max_{y \in \partial q} -\|y\|_2,$$

which yields the above program (*cf*. [2], Chapter 4). If the zero vector is part of the subdifferential, we are dual optimal. Primal optimality follows from Claim 1. ☐

One can monotonically decrease the dual objective by minimizing it along the steepest descent direction. Unfortunately, following the steepest descent direction does not guarantee convergence to the global minimum of the dual function [28]. Performing steepest descent might keep minimizing the dual objective with smaller and smaller increments, thus converging to a suboptimal solution. The main drawback of steepest descent as well as block coordinate descent when applied to the dual objective in Eq. (3) is that both procedures only consider the support of $X_i^*, X_\alpha^*$ defined in Claim 1. In the following we show that by considering the $\epsilon$-margin of these supports we can guarantee that at every iteration we decrease the dual value by at least $\epsilon$. This procedure results in an efficient algorithm that reaches both dual and primal optimal solutions.

## 4    The $\epsilon$-Subgradients of the Dual Objective and Steepest $\epsilon$-Descent

To monotonically decrease the dual value while converging to the optimum, we suggest to explore the $\epsilon$-neighborhood of the dual objective in Eq. (3) around the current iterate $\lambda$. For this purpose, we explore its family of $\epsilon$-subgradients. Given our convex dual function $q(\lambda)$ and a positive scalar $\epsilon$, we say that a vector $d$ is an $\epsilon$-subgradient at $\lambda$ if it supports the epigraph of $q(\lambda)$ with an $\epsilon$-margin:

$$\forall \hat{\lambda} \qquad q(\hat{\lambda}) - d^\top \hat{\lambda} \geq q(\lambda) - d^\top \lambda - \epsilon. \tag{5}$$

The subgradients of a convex function are also $\epsilon$-subgradients. The family of $\epsilon$-subgradients is called the $\epsilon$-subdifferential and is denoted by $\partial_\epsilon q(\lambda)$. Using the conjugate dual $q^*(d)$, we can characterize the $\epsilon$-subdifferential by employing the $\epsilon$-margin Fenchel-Young duality theorem.

$$(\epsilon\text{-margin Fenchel-Young duality}) \qquad \partial_\epsilon q(\lambda) = \left\{ d : 0 \leq q(\lambda) + q^*(d) - d^\top \lambda \leq \epsilon \right\} \tag{6}$$

The $\epsilon$-subdifferential augments the subdifferential of $q(\lambda)$ with additional directions $d$ which control the $\epsilon$-neighborhood of the function. Whenever one finds a steepest descent direction within $\partial_\epsilon q(\lambda)$, it is guaranteed to improve the dual objective by at least $\epsilon$. Moreover, if one cannot find such a direction within the $\epsilon$-subdifferential, then $q(\lambda)$ is guaranteed to be $\epsilon$-close to the dual optimum. This is summarized in the following claim.

**Claim 2.** *Let $q(\lambda)$ be a convex function and let $\epsilon$ be a positive scalar. The $\epsilon$-subdifferential $\partial_\epsilon q(\lambda)$ is a convex and compact set. If $0 \notin \partial_\epsilon q(\lambda)$ then the direction $d^* = \operatorname{argmin} \|d\|$ subject to $d \in \partial_\epsilon q(\lambda)$ is a descent direction and $\inf_{\alpha > 0} q(\lambda - \alpha d) < q(\lambda) - \epsilon$. On the other hand, if $0 \in \partial_\epsilon q(\lambda)$ then $q(\lambda) \leq \inf_\lambda q(\lambda) + \epsilon$.*

**Proof:** [2] Proposition 4.3.1. ☐

Although $\partial_\epsilon q(\lambda)$ is a convex and compact set, finding its direction of descent is computationally challenging. Fortunately, it can be approximated whenever the convex function is a sum of simple convex functions, *i.e.*, $q(\lambda) = \sum_{r=1}^m q_r(\lambda)$. The approximation $\tilde{\partial}_\epsilon q(\lambda) = \sum_r \partial_\epsilon q_r(\lambda)$ satisfies $\partial_\epsilon q(\lambda) \subset \tilde{\partial}_\epsilon q(\lambda) \subset \partial_{m\epsilon} q(\lambda)$, (see, *e.g.*, [2]). On the one hand, if $0 \notin \tilde{\partial}_\epsilon q(\lambda)$ then the direction of steepest descent taken from $\tilde{\partial}_\epsilon q(\lambda)$ reduces the dual objective by at least $\epsilon$. If $0 \in \tilde{\partial}_\epsilon q(\lambda)$ then $q(\lambda)$ is $m\epsilon$-close to the dual optimum. In the following claim we use the $\epsilon$-margin Fenchel-Young duality in Eq. (6) to characterize the approximated $\epsilon$-subdifferential of the dual function.

**Claim 3.** *Consider the dual function $q(\lambda)$ in Eq. (3). Then the approximated $\epsilon$-subdifferential consists of vectors $d$ whose entries correspond to marginalization disagreements,* i.e., *$d \in \tilde{\partial}_\epsilon q(\lambda)$ if and only if $d_{i\to\alpha}(x_i) = \sum_{x_\alpha \setminus x_i} b_\alpha(x_\alpha) - b_i(x_i)$ for probability distributions $b_i(x_i), b_\alpha(x_\alpha)$ that satisfy*

$$\forall i \quad \max_{x_i} \left\{ \theta_i(x_i) - \sum_{\alpha \in N(i)} \lambda_{i\to\alpha}(x_i) \right\} - \epsilon \leq \sum_{x_i} b_i(x_i) \left( \theta_i(x_i) - \sum_{\alpha \in N(i)} \lambda_{i\to\alpha}(x_i) \right)$$

$$\forall \alpha \quad \max_{x_\alpha} \left\{ \theta_\alpha(x_\alpha) + \sum_{i \in N(\alpha)} \lambda_{i\to\alpha}(x_i) \right\} - \epsilon \leq \sum_{x_\alpha} b_\alpha(x_\alpha) \left( \theta_\alpha(x_\alpha) + \sum_{i \in N(\alpha)} \lambda_{i\to\alpha}(x_i) \right).$$

**Proof:** Eq. (6) implies $b \in \partial_\epsilon q_r(\hat{\theta})$ if and only if $q_r(\hat{\theta}) + q_r^*(b) - b^\top \hat{\theta} \leq \epsilon$ with $q_r^*(b)$ denoting the conjugate dual of $q_r(\hat{\theta})$. Plugging in $q_r, q_r^*$ we obtain not only the maximizing beliefs but all beliefs with an $\epsilon$-margin. Summing over $r \in \{i, \alpha\}$ while noticing that $\lambda_{i\to\alpha}(x_i)$ change signs between $q_\alpha$ and $q_i$ we obtain the marginalization disagreements $d_{i\to\alpha}(x_i) = \sum_{x_\alpha \setminus x_i} b_\alpha(x_\alpha) - b_i(x_i)$. $\square$

$\tilde{\partial}_\epsilon q(\lambda)$ is described using beliefs $b_i(x_i), b_\alpha(x_\alpha)$ that satisfy linear constraints, therefore finding a direction of $\epsilon$-descent can be done efficiently. Claim 2 ensures that minimizing the dual objective along a direction of descent decreases its value by at least $\epsilon$. Moreover, we are guaranteed to be $(|V|+|E|)\epsilon$-close to a dual optimal solution if no direction of descent is found in $\tilde{\partial}_\epsilon q(\lambda)$. Therefore, we are able to get out of corners and efficiently reach an approximated dual optimal solution. The interpretation of the Fenchel-Young margin as the amount of disagreement between the marginalization constraints also provides a simple way to reconstruct an approximately optimal primal solution. This is summarized in the following corollary.

**Corollary 2.** *Given a point $\lambda$, set $\hat{\theta}_i(x_i) = \theta_i(x_i) - \sum_{\alpha \in N(i)} \lambda_{i\to\alpha}(x_i)$ and $\hat{\theta}_\alpha(x_\alpha) = \theta_\alpha(x_\alpha) + \sum_{i \in N(\alpha)} \lambda_{i\to\alpha}(x_i)$. Consider the quadratic program*

$$\min_{b_i, b_\alpha} \sum_{i, x_i, \alpha \in N(i)} \left( \sum_{x_\alpha \setminus x_i} b_\alpha(x_\alpha) - b_i(x_i) \right)^2$$

$$s.t. \quad b_i(x_i), b_\alpha(x_\alpha) \geq 0, \quad \sum_{x_\alpha} b_\alpha(x_\alpha) = 1, \sum_{x_i} b_i(x_i) = 1$$

$$\sum_{x_i} b_i(x_i)\hat{\theta}_i(x_i) \geq \max_{x_i}\{\hat{\theta}_i(x_i)\} - \epsilon, \quad \sum_{x_\alpha} b_\alpha(x_\alpha)\hat{\theta}_\alpha(x_\alpha) \geq \max_{x_\alpha}\{\hat{\theta}_\alpha(x_\alpha)\} - \epsilon.$$

*$q(\lambda)$ is $(|V| + |E|)\epsilon$-close to the dual optimal value if and only if the value of the above program equals zero. Moreover, the optimal beliefs $b_\alpha^*(x_\alpha), b_i^*(x_i)$ primal value is $(|V| + |E|)\epsilon$-close to the optimal primal value in Eq. (2). However, if $q(\lambda)$ is not $(|V|+|E|)\epsilon$-close to the dual optimal value then the vector $d_{i\to\alpha}^*(x_i) = \sum_{x_\alpha \setminus x_i} b_\alpha^*(x_\alpha) - b_i^*(x_i)$ points towards the steepest $\epsilon$-descent direction of the function, namely*

$$d^* = \operatorname*{argmin}_{\|d\| \leq 1} \lim_{\alpha \downarrow 0} \frac{q(\lambda + \alpha d) - q(\lambda) + \epsilon}{\alpha}.$$

**Proof:** The steepest $\epsilon$-descent direction is given by the minimum norm element of the $\epsilon$-subdifferential, described in Claim 3. $(|V| + |E|)\epsilon$-closeness to the dual optimum is given by ([2], Proposition 4.3.1) once we find the value of the quadratic program to be zero. Note that the superset $\tilde{\partial}_\epsilon$ is composed of $|V| + |E|$ subdifferentials. If the value of the above program equals zero, the beliefs fulfill marginalization constraints and they denote a probability distribution. Summing both $\epsilon$-margin inequalities w.r.t. $i, \alpha$, we obtain

$$\sum_{i,x_i} b_i(x_i)\hat{\theta}_i(x_i) + \sum_{\alpha,x_\alpha} b_\alpha(x_\alpha)\hat{\theta}_\alpha(x_\alpha) \geq \sum_i \max_{x_i} \hat{\theta}_i(x_i) + \sum_\alpha \max_{x_\alpha} \hat{\theta}_\alpha(x_\alpha) - (|V| + |E|)\epsilon.$$

where the primal on the left hand side of the resulting inequality is larger then the dual subtracted by $(|V| + |E|)\epsilon$. With the dual itself upper bounding the primal, the corollary follows. $\square$

Thus, we can construct an algorithm that performs $\epsilon$ improvements over the dual function in each iteration. We can either perform block-coordinate dual descent (*i.e.*, convex max-product updates) or steepest $\epsilon$-descent steps. Since both methods monotonically improve the same dual function, our approach is guaranteed to reach the optimal dual solution and to recover the primal optimal solution.

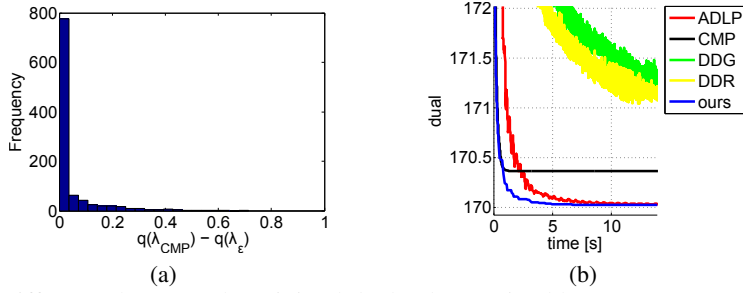

Figure 1: (a) Difference between the minimal dual value attained by convex max-product $q(\lambda_{\text{CMP}})$ and our approach $q(\lambda_\epsilon)$. Convex max-product gets stuck in about $20\%$ of all cases. (b) Dual value achieved after a certain amount of time for cases where convex max-product gets stuck.

## 5 High-Order Region Graphs

Graphical models naturally describe probability distributions with different types of regions $r \subset \{1, ..., n\}$. However, the linear program relaxation described in Eq. (2) considers interactions between regions which correspond to variables $i$ and regions that correspond to cliques $\alpha$. In the following we extend the $\epsilon$-descent framework when considering linear programming relaxations without constraining the region interactions. Since we allow any regions to interact, we denote these interactions through a region graph [29]. A region graph is a directed graph whose nodes represent the regions and its direct edges correspond to the inclusion relation, *i.e.*, a directed edge from node $r$ to $s$ is possible only if $s \subset r$. We adopt the terminology where $P(r)$ and $C(r)$ stand for all nodes that are parents and children of the node $r$, respectively. Thus we consider the linear programming relaxation of a general high-order graphical model as follows

$$\max_b \quad \sum_{r, x_r} b_r(x_r)\theta_r(x_r) \tag{7}$$

$$\text{s.t.} \quad b_r(x_r) \geq 0, \quad \sum_{x_r} b_r(x_r) = 1, \quad \forall r, s \in P(r) \sum_{x_s \backslash x_r} b_s(x_s) = b_r(x_r)$$

Following [5, 22, 27] we consider the re-parametrized dual program

$$q(\lambda) = \sum_r \max_{x_r} \left\{ \theta_r(x_r) + \sum_{c \in C(r)} \lambda_{c \to r}(x_c) - \sum_{p \in P(r)} \lambda_{r \to p}(x_r) \right\}$$

which is a sum of max-functions. Its approximated $\epsilon$-subdifferential is described with respect to their Fenchel-Young margins. Using the same reasoning as in Sec. 4 we present a simple way to recover an $\epsilon$-steepest descent direction, as well as to reconstruct an approximated optimal primal solution.

**Corollary 3.** *Given a point $\lambda$, set $\hat{\theta}_r(x_r) = \theta_r(x_r) + \sum_{c \in C(r)} \lambda_{c \to r}(x_r) - \sum_{p \in P(r)} \lambda_{r \to p}(x_r)$. Consider the quadratic program*

$$\min_b \quad \sum_{r, x_r, p \in P(r)} \left( \sum_{x_p \backslash x_r} b_p(x_p) - b_r(x_r) \right)^2$$

$$\text{s.t.} \quad b_r(x_r) \geq 0, \quad \sum_{x_r} b_r(x_r) = 1, \quad \sum_{x_r} b_r(x_r)\hat{\theta}_r(x_r) \geq \max_{x_r}\{\hat{\theta}_r(x_r)\} - \epsilon$$

*Let $|R|$ be the total number of regions in the graph, then $\lambda$ is $|R|\epsilon$-close to the dual optimal solution if and only if the value of the above program equals zero. Moreover, the optimal beliefs $b_r^*(x_r)$ are also $|R|\epsilon$-close to the optimal solution of the primal program in Eq. (7). However, if $q(\lambda)$ is not $|R|\epsilon$ close to the dual optimal solution then the vector $d_{r \to p}^*(x_r) = \sum_{x_p \backslash x_r} b_p^*(x_p) - b_r^*(x_r)$ points towards the steepest $\epsilon$-descent direction of the dual function.*

**Proof:** It is a straightforward generalization of Corollary 2. □

When dealing with high-order region graphs, one can choose a region graph, *e.g.*, the Hasse diagram, that has significantly less edges than a region graph that connects variables $i$ to cliques $\alpha$. Therefore, when considering many high-order regions, the formulation in the above corollary is more efficient than the one in Corollary 2.

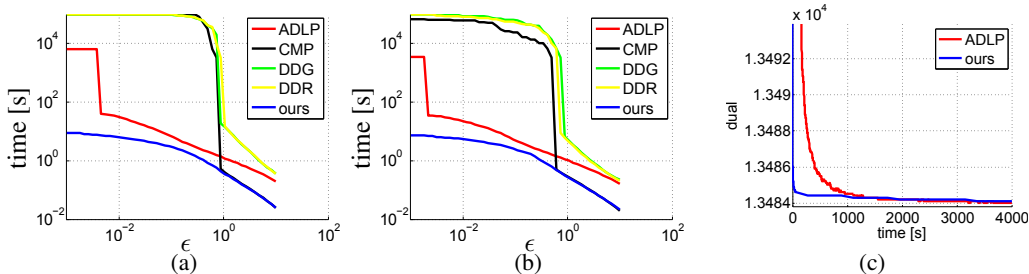

Figure 2: Average time required for different solvers to achieve a specified accuracy on 30 spin glass models, (a) when solvers are applied to "hard" problems only, *i.e.*, those where CMP gets stuck far from the optimum. Average results over 30 models are shown in (b), (c) decrease of the dual value over time for ADLP and our $\epsilon$-descent approach.

## 6 Experimental Evaluation

To benefit from the efficiency of convex max-product, our implementation starts by employing block-coordinate descent iterations before switching to the globally convergent $\epsilon$-descent approach once the dual value decreases by less than $\epsilon = 0.01$. As we always optimize the same cost function, switching the gradient computation is possible. We employ a backtracking line search in our $\epsilon$-descent approach. In the following we demonstrate the effectiveness of our approach on synthetic 10x10 spin glass models as well as protein interactions from the probabilistic inference challenge (PIC 2011). We consider **spin glass models** that consist of local factors, each having 3 states with values randomly chosen according to $\mathcal{N}(0,1)$. We use three states as convex max-product is optimal for pairwise spin glass models with only two states per random variable. The pairwise factors of the regular grid are weighted potentials with $+1$ on the diagonal and off-diagonal entries being $-1$. The weights are again independently drawn from $\mathcal{N}(0,1)$. In the first experiment we are interested in estimating how often convex max-product gets stuck in corners. We generate a set of 1000 spin glass models and estimate the distribution of the dual value difference comparing the $\epsilon$-descent approach with the convex max-product result after $10,000$ iterations. We observe in Fig. 1(a) that about $20\%$ of the spin glass models have a dual value difference larger than zero.

Having observed that convex max-product does not achieve optimality for $20\%$ of the models, we now turn our attention to evaluating the run-time of different algorithms. We compare our implementation of the $\epsilon$-steepest descent algorithm with the alternating direction method for dual MAP-LP relaxations (ADLP) [18]. In addition, we illustrate the performance of convex max-product (CMP) [6] and compare against the dual-decomposition work of [12] provided in a generic (DDG) and a re-weighted (DDR) version in the STAIR library [4]. Note that ADLP is also implemented in this library. All algorithms are restricted to at most $20,000$ iterations. We draw the readers attention to Fig. 1(b), where we evaluate a single spin glass model and illustrate the dual value obtained after a certain amount of time. As given by the derivations, CMP is a monotonically decreasing algorithm that can get stuck in corners. It is important to note that our $\epsilon$-descent approach is monotonically decreasing as well, which contrasts all the other investigated algorithms (ADLP, DDG, DDR).

We evaluate the time it requires the different algorithms to achieve a given accuracy. We first focus on "hard" problems, where we defined "hard" as those spin glass models whose difference between convex max-product and the $\epsilon$-descent method is larger than $0.2$. To obtain statistically meaningful results we average over 30 hard problems and report the time to achieve a given accuracy in Fig. 2(a). We used the minimum across all dual values found by all algorithms as the optimum. If an algorithm does not achieve $\epsilon$-close accuracy within 20,000 iterations we set its time to the arbitrarily chosen value of $10^5$. We note that CMP is very fast for low accuracies (high $\epsilon$) but gets stuck in corners, not achieving high accuracies (low $\epsilon$). This is also the case for DDG and DDR. ADLP achieves significantly lower $\epsilon$-closeness but the $20,000$ iteration limit stops it from reaching $10^{-3}$. The previous experiment focus on hard problems. In order to evaluate the average case, we randomly generate 30 spin glass models. The results are provided in Fig. 2(b). As expected the $\epsilon$-descent approach performs similarly well, ADLP achieves lower accuracies on more samples. The step apparent for CMP, DDG and DDR is not as sharp, but still very significant.

**Protein interactions**: We rely on the data provided by the PIC 2011 and compare the $\epsilon$-descent approach to ADLP as it is the most competitive method in the previous experiments. The dual energy obtained after a given amount of time is illustrated in Fig. 2(c).

# 7 Related Work

We explore methods to solve LP relaxations by monotonically decreasing the value of its dual objective and reconstructing a primal optimal solution. For this purpose we investigate approximated subgradients of the dual program using the Fenchel-Young margins, and provide a method to reduce the dual objective in every step by a constant value until convergence to the optimum. Efficient dual solvers were extensively studied in the context of LP relaxations for the MAP problem [14, 20, 25]. The dual program is non-smooth, thus subgradient descent algorithms are guaranteed to reach the dual optimum [12], as well as recover the primal optimum [12]. Despite their theoretical guarantees, subgradient methods are typically slow. Dual block coordinate descent methods, typically referred to as convex max-product algorithms, are monotonically decreasing, and were shown to be faster than subgradient methods [3, 6, 11, 17, 22, 24, 27]. Since the dual program is non-smooth, these algorithms can get stuck in non-optimal stationary points and cannot in general recover a primal optimal solution [26]. Our work specifically addresses these drawbacks.

Recently, several methods were devised to overcome the sub-optimality of convex max-product algorithms. Unlike our approach, all these algorithms optimize a perturbed program. Some methods use the soft-max with low temperature to smooth the dual objective in order to avoid corners as well as to recover primal optimal solutions [6, 7, 8]. However, these methods are typically slower, as computation of the low-temperature soft-max is more expensive than max-computation. [19] applied the proximal method, employing a primal strictly concave perturbation, which results in a smooth dual approximation that is temperature independent. This approach converges to the dual optimum and recovers the primal optimal solution. However, it uses a double loop scheme where every update involves executing a convex sum-product algorithm. Alternative methods applied augmented Lagrangian techniques to the primal [16] and the dual programs [18]. The augmented Lagrangian method guarantees to reach the global optimum and recover the dual and primal solutions. Unlike our approach, this method is not monotonically decreasing and works on a perturbed objective, thus cannot be efficiently integrated with convex max-product updates that perform block coordinate descent on the dual of the LP relaxation.

Our approach is based on the $\epsilon$-descent algorithm for convex functions [2]. We use the $\epsilon$-margin of the Fenchel-Young duality theorem to adjust the $\epsilon$-subdifferential to the dual objective of the LP relaxation, thus augmenting the convex max-product with the ability to get out of corners. We also construct an efficient method to recover a primal optimal solution. Our approach is related to the Bundle method [15, 9], which performs an $\epsilon$-subgradient descent in cases where efficient search in the $\epsilon$-subdifferential is impossible. The graphical model structure in our setting makes searching in the $\epsilon$-subdifferential easy, thus our approach is significantly faster. Our algorithm satisfies $\epsilon$-complementary slackness while performing $\epsilon$-descent step, similarly to the auction algorithm. However, our algorithm is monotonically decreasing and can be used for general graphical models, while the auction algorithm might increase its dual and its convergence properties hold only for network flow problems.

# 8 Conclusions and Discussion

Evaluating the MAP assignment and solving its LP relaxations are key problems in approximate inference. Some of the existing solvers, such as convex max-product, have limitations. Mainly, these solvers can get stuck in a non-optimal stationary point, thus they cannot recover the primal optimal solution. We explore the properties of subgradients of the dual objective and construct a simple algorithm that determines if the dual stationary point is optimal and recovers the primal optimal solution in this case (Corollary 1). Moreover, we investigate the family of $\epsilon$-subgradients using Fenchel-Young margins and construct a monotonically decreasing algorithm that is guaranteed to achieve optimal dual and primal solutions (Corollary 2), including general region graphs (Corollary 3). We show that our algorithm compares favorably with pervious methods on spin glass models and protein interactions. The approximated steepest descent direction is recovered by solving a quadratic program subject to linear constraints. We used the Gurobi solver[2], which ignores the graphical structure of the linear constraints. We believe that constructing a message-passing solver for this sub-problem will significantly speed-up our approach. Further extensions, e.g., enforcing constraints over messages such as those arising from cloud computing are also applicable to our setting [1, 21].

## Footnotes

[2]http://www.gurobi.com

# References

[1] A. Auslender and M. Teboulle. Interior gradient and epsilon-subgradient descent methods for constrained convex minimization. *Mathematics of Operations Research*, 2004.

[2] D. P. Bertsekas, A. Nedić, and A. E. Ozdaglar. *Convex Analysis and Optimization*. Athena Scientific, 2003.

[3] A. Globerson and T. S. Jaakkola. Fixing max-product: convergent message passing algorithms for MAP relaxations. In *Proc. NIPS*, 2007.

[4] S. Gould, O. Russakovsky, I. Goodfellow, P. Baumstarck, A. Y. Ng, and D. Koller. The STAIR Vision Library (v2.4), 2011. http://ai.stanford.edu/ sgould/svl.

[5] T. Hazan, J. Peng, and A. Shashua. Tightening fractional covering upper bounds on the partition function for high-order region graphs. In *Proc. UAI*, 2012.

[6] T. Hazan and A. Shashua. Norm-product belief propagation: Primal-dual message-passing for approximate inference. *Trans. on Information Theory*, 2010.

[7] J. K. Johnson. *Convex relaxation methods for graphical models: Lagrangian and maximum entropy approaches*. PhD thesis, Massachusetts Institute of Technology, 2008.

[8] V. Jojic, S. Gould, and D. Koller. Accelerated dual decomposition for MAP inference. In *Proc. ICML*, 2010.

[9] J. H. Kappes, B. Savchynskyy, and C. Schnörr. A Bundle Approach To Efficient MAP-Inference by Lagrangian Relaxation. In *Proc. CVPR*, 2012.

[10] D. Koller and N. Friedman. *Probabilistic graphical models*. MIT Press, 2009.

[11] V. Kolmogorov. Convergent tree-reweighted message passing for energy minimization. *PAMI*, 2006.

[12] N. Komodakis, N. Paragios, and G. Tziritas. MRF Energy Minimization & Beyond via Dual Decomposition. *PAMI*, 2010.

[13] T. Koo, A.M. Rush, M. Collins, T. Jaakkola, and D. Sontag. Dual decomposition for parsing with non-projective head automata. In *Proc. EMNLP*, 2010.

[14] A.M.C.A. Koster, S.P.M. van Hoesel, and A.W.J. Kolen. The partial constraint satisfaction problem: Facets and lifting theorems. *Operations Research Letters*, 1998.

[15] C. Lemaréchal. An algorithm for minimizing convex functions. *Information processing*, 1974.

[16] A.F.T. Martins, M.A.T. Figueiredo, P.M.Q. Aguiar, N.A. Smith, and E.P. Xing. An Augmented Lagrangian Approach to Constrained MAP Inference. In *Proc. ICML*, 2011.

[17] T. Meltzer, A. Globerson, and Y. Weiss. Convergent Message Passing Algorithms – A Unifying View. In *Proc. UAI*, 2009.

[18] O. Meshi and A. Globerson. An Alternating Direction Method for Dual MAP LP Relaxation. In *Proc. ECML PKDD*, 2011.

[19] P. Ravikumar, A. Agarwal, and M. J. Wainwright. Message-passing for graph-structured linear programs: Proximal methods and rounding schemes. *JMLR*, 2010.

[20] M. Schlesinger. Syntactic analysis of two-dimensional visual signals in noisy conditions. *Kibernetika,76*.

[21] A. G. Schwing, T. Hazan, M. Pollefeys, and R. Urtasun. Distributed message passing for large scale graphical models. In *Proc. CVPR*, 2011.

[22] D. Sontag and T. S. Jaakkola. Tree block coordinate descent for MAP in graphical models. In *Proc. AISTATS*, 2009.

[23] D. Sontag, T. Meltzer, A. Globerson, T. Jaakkola, and Y. Weiss. Tightening LP relaxations for MAP using message passing. In *Proc. UAI*, 2008.

[24] D. Tarlow, D. Batra, P. Kohli, and V. Kolmogorov. Dynamic tree block coordinate ascent. In *Proc. ICML*, 2011.

[25] M. J. Wainwright, T. S. Jaakkola, and A. S. Willsky. MAP estimation via agreement on trees: message-passing and linear programming. *Trans. on Information Theory*, 2005.

[26] Y. Weiss, C. Yanover, and T. Meltzer. MAP Estimation, Linear Programming and Belief Propagation with Convex Free Energies. In *Proc. UAI*, 2007.

[27] T. Werner. Revisiting the linear programming relaxation approach to gibbs energy minimization and weighted constraint satisfaction. *PAMI*, 2010.

[28] P. Wolfe. A method of conjugate subgradients for minimizing nondifferentiable functions. *Nondifferentiable Optimization*, 1975.

[29] J. S. Yedidia, W. T. Freeman, and Y. Weiss. Constructing free-energy approximations and generalized belief propagation algorithms. *Trans. on Information Theory*, 2005.

